# Streaming Pointwise Mutual Information

**Benjamin Van Durme**
University of Rochester
Rochester, NY 14627, USA

**Ashwin Lall**
Georgia Institute of Technology
Atlanta, GA 30332, USA

## Abstract

Recent work has led to the ability to perform space efficient, approximate counting over large vocabularies in a streaming context. Motivated by the existence of data structures of this type, we explore the computation of associativity scores, otherwise known as *pointwise mutual information* (PMI), in a streaming context. We give theoretical bounds showing the impracticality of perfect online PMI computation, and detail an algorithm with high *expected* accuracy. Experiments on news articles show our approach gives high accuracy on real world data.

## 1 Introduction

Recent work has led to the ability to perform space efficient counting over large vocabularies [Talbot, 2009; Van Durme and Lall, 2009]. As online extensions to previous work in randomized storage [Talbot and Osborne, 2007], significant space savings are enabled if your application can tolerate a small chance of false positive in lookup, and you do not require the ability to enumerate the contents of your collection.[1] Recent interest in this area is motivated by the scale of available data outpacing the computational resources typically at hand.

We explore what a data structure of this type means for the computation of associativity scores, or pointwise mutual information, in a streaming context. We show that approximate $k$-best PMI rank lists may be maintained online, with high accuracy, both in theory and in practice. This result is useful both when storage constraints prohibit explicitly storing all observed co-occurrences in a stream, as well as in cases where accessing such PMI values would be useful online.

## 2 Problem Definition and Notation

Throughout this paper we will assume our data is in the form of pairs $\langle x, y \rangle$, where $x \in X$ and $y \in Y$. Further, we assume that the sets $X$ and $Y$ are so large that it is infeasible to explicitly maintain precise counts for every such pair on a single machine (e.g., $X$ and $Y$ are all the words in the English language).

We define the pointwise mutual information (PMI) of a pair $x$ and $y$ to be

$$\text{PMI}(x, y) \equiv \lg \frac{P(x, y)}{P(x)P(y)}$$

where these (empirical) probabilities are computed over a particular data set of interest.[2] Now, it is often the case that we are not interested in *all* such pairs, but instead are satisfied with estimating the subset of $Y$ with the $k$ largest PMIs with each $x \in X$. We denote this set by $\text{PMI}_k(x)$.

Our goal in this paper is to estimate these top-$k$ sets in a streaming fashion, i.e., where there is only a single pass allowed over the data and it is infeasible to store all the data for random access. This

model is natural for a variety of reasons, e.g., the data is being accessed by crawling the web and it is infeasible to buffer all the crawled results.

As mentioned earlier, there has been considerable work in keeping track of the counts of a large number of items succinctly. We explore the possibility of using these succinct data structures to solve this problem. Suppose there is a multi-set $M = \{m_1, m_2, m_3, \ldots\}$ of word pairs from $X \times Y$. Using an approximate counter data structure, it is possible to maintain in an online fashion the counts

$$
\begin{aligned}
c(x,y) &= |\{i \mid m_i = \langle x, y \rangle\}|, \\
c(x) &= |\{i \mid m_i = \langle x, y' \rangle, \text{for some } y' \in Y\}|, \text{ and} \\
c(y) &= |\{i \mid m_i = \langle x', y \rangle, \text{for some } x' \in X\}|,
\end{aligned}
$$

which allows us to estimate $\text{PMI}(x,y)$ as $\lg \frac{P(x,y)}{P(x)P(y)} = \lg \frac{c(x,y)/n}{(c(x)/n)(c(y)/n)} = \lg \frac{nc(x,y)}{c(x)c(y)}$, where $n$ is the length of the stream. The challenge for this problem is determining how to keep track of the set $\text{PMI}_k(x)$ for all $x \in X$ in an online fashion.

## 3 Motivation

Pointwise mutual information underlies many experiments in computational (psycho-)linguistics, going back at least to Church and Hanks [1990], who at the time referred to PMI as a mathematical formalization of the psycholinguistic *association score*. We do not attempt to summarize this work in its entirety, but give representative highlights below.

**Trigger Models** Rosenfeld [1994] was interested in collecting *trigger pairs*, $\langle A, B \rangle$, such that the presence of $A$ in a document is likely to "trigger" an occurrence of $B$. There the concern was in finding the most useful triggers overall, and thus pairs were favored based on high *average* mutual information; $I(A, B) = P(AB) \lg \frac{P(AB)}{P(A)P(B)} + P(A\bar{B}) \lg \frac{P(A\bar{B})}{P(A)P(\bar{B})} + P(\bar{A}\bar{B}) \lg \frac{P(\bar{A}\bar{B})}{P(\bar{A})P(\bar{B})} + P(\bar{A}B) \lg \frac{P(\bar{A}B)}{P(\bar{A})P(B)}$.

As commented by Rosenfeld, the first term of his equation relates to the PMI formula given by Church and Hanks [1990]. We might describe our work here as collecting terms $y$, triggered by each $x$, once we know $x$ to be present. As the number of possible terms is large,[3] we limit ourselves to the top-$k$ items.

**Associated Verbs** Chambers and Jurafsky [2008], following work such as Lin [1998] and Chklovski and Pantel [2004], introduced a probabilistic model for learning Shankian script-like structures which they termed *narrative event chains*; for example, if in a given document someone *pleaded*, *admits* and was *convicted*, then it is likely they were also *sentenced*, or *paroled*, or *fired*. Prior to enforcing a temporal ordering (which does not concern us here), Chambers and Jurafsky acquired clusters of related verb-argument pairs by finding those that shared high PMI.

**Associativity in Human Memory** Central to their rational analysis of human memory, Schooler and Anderson [1997] approximated the *needs odds*, $n$, of a memory structure $S$ as the product of *recency* and *context* factors, where the context factor is the product of *associative ratios* between $S$ and local *cues*; $n \cong \frac{P(S|H_S)}{P(\bar{S}|H_S)} \prod_{q \in Q_S} \frac{P(Sq)}{P(S)P(q)}$.

If we take $x$ to range over cues, and $y$ to be a memory structure, then in our work here we are storing the identities of the top-$k$ memory structures for a given cue $x$, as according to strength of associativity.[4]

## 4 Lower Bound

We first discuss the difficulty in solving the online PMI problem exactly. An obvious first attempt at an algorithm for this problem is to use approximate counters to estimate the PMI for each pair in

the stream and maintain the top-$k$ for each $x$ using a priority queue. This method does not work, as illustrated by the examples below.

**Example 1 (probability of $y$ changes)**: Consider the stream

$$xy \; xy \; xy \; xz \; wz \mid wy \; wy \; wy \; wy \; wy$$

which we have divided in half. After the first half, $y$ is best for $x$ since $\mathrm{PMI}(x, y) = \lg \frac{3/5}{(4/5)(3/5)} = \lg (5/4)$ and $\mathrm{PMI}(x, z) = \lg \frac{1/5}{(4/5)(2/5)} = \lg (5/8)$. At the end of the second half of the stream, $z$ is best for $x$ since $\mathrm{PMI}(x, y) = \lg \frac{3/10}{(4/10)(8/10)} \approx \lg (0.94)$ and $\mathrm{PMI}(x, z) = \lg \frac{1/10}{(4/10)(2/10)} = \lg (1.25)$. However, during the second half of the stream we never encounter $x$ and hence never update its value. So, the naive algorithm behaves erroneously.

What this example shows is that not only does the naive algorithm fail, but also that the top-$k$ PMI of some $x$ may change (because of the change in probability of $y$) without any opportunity to update $\mathrm{PMI}_k(x)$.

Next, we show another example which illustrates the failure of the naive algorithm due to the fact that it does not re-compute every PMI each time.

**Example 2 (probability of $x$ changes)**: Consider the stream

$$pd \; py \; py \; xy \; xd$$

in which we are interested in only the top PMI tuples for $x$. When we see $xy$ in the stream, $\mathrm{PMI}(x, y) = \lg \frac{1/4}{(1/4)(3/4)} \approx \lg (1.33)$, and when we see $xd$ in the stream, $\mathrm{PMI}(x, d) = \lg \frac{1/5}{(2/5)(2/5)} = \lg (1.25)$. As a result, we retain $xy$ but not $xd$. However, $xy$'s PMI is now $\mathrm{PMI}(x, y) = \lg \frac{1/5}{(2/5)(3/5)} = \lg (0.833)$ which means that we should replace $xy$ with $xd$. However, since we didn't re-compute $\mathrm{PMI}(x, y)$, we erroneously output $xy$.

We next formalize these intuitions into a lower bound showing why it might be hard to compute every $\mathrm{PMI}_k(x)$ precisely. For this lower bound, we make the simplifying assumption that the size of the set $X$ is much smaller than $N$ (i.e., $|X| \in o(N)$), which is the usual case in practice.

**Theorem 1:** Any algorithm that explicitly maintains the top-$k$ PMIs for all $x \in X$ in a stream of length at most $n$ (where $|X| \in o(n)$) in a single pass requires $\Omega(n|X|)$ time.

We will prove this theorem using the following lemma:

**Lemma 1:** Any algorithm that explicitly maintains the top-$k$ PMIs of $|X| = p + 1$ items over a stream of length at most $n = 2r + 2p + 1$ in a single pass requires $\Omega(pr)$ time.

**Proof of Lemma 1:** Let us take the length of the stream to be $n$, where we assume without loss of generality that $n$ is odd. Let $X = \{x_1, \ldots, x_{p+1}\}$, $Y = \{y_1, y_2\}$ and let us consider the following stream:

$$
\begin{aligned}
& x_1 y_1, \; x_2 y_1, \; x_3 y_1, \; \ldots, \; x_p y_1, \\
& x_1 y_2, \; x_2 y_2, \; x_3 y_2, \; \ldots, \; x_p y_2, \\
& \hspace{5cm} x_{p+1} y_1
\end{aligned}
$$

$$
\left.
\begin{aligned}
& x_{p+1} y_2, \quad x_{p+1} y_2, \\
& x_{p+1} y_1, \quad x_{p+1} y_1, \\
& x_{p+1} y_2, \quad x_{p+1} y_2, \\
& \quad\quad \ldots \\
& x_{p+1} y_{1+r(mod)2}, \quad x_{p+1} y_{1+r(mod)2}.
\end{aligned}
\right\} r \text{ times}
$$

Suppose that we are interested in maintaining only the top-PMI item for each $x_i \in X$ (the proof easily generalizes to larger $k$). Let us consider the update cost for only the set $X_p = \{x_1, \ldots, x_p\} \subseteq X$. After $x_{p+1} y_1$ appears in the stream for the first time, it should be evident that all the elements of $X_p$ have a higher PMI with $y_2$ than $y_1$. However, after we see two copies of $x_{p+1} y_2$, the PMI of $y_1$ is higher than that of $y_2$ for each $x \in X_p$. Similarly, the top-PMI of each element of $X_p$ alternates between $y_1$ and $y_2$ for the remainder of the stream. Now, the current PMI for each element of $X_p$ must be correct at any point in the stream since the stream may terminate at any time. Hence, by construction, the top PMI of $x_1, \ldots, x_p$ will change at least $r$ times in the course of this stream, for

a total of at least $pr$ operations. The length of the stream is $n = 2p + 2r + 1$. This completes the proof of Lemma 1. $\square$

**Proof of Theorem 1:** Taking $|X| = p + 1$, we have in the construction of Lemma 1 that $r = (n - 2p - 1)/2 = (n - 2|X| + 1)/2$. Hence, there are at least $pr = (|X| - 1)(n - 2|X| + 1)/2 = \Omega(n|X| - |X|^2)$ update operations required. Since we assumed that $|X| \in o(n)$, this is $\Omega(n|X|)$ operations. $\square$

Hence, there must be a high update cost for any such algorithm. That is, on average, any algorithm must perform $\Omega(|X|)$ operations per item in the stream.

# 5   Algorithm

The lower bound from the previous section shows that, when solving the PMI problem, the best one can do is effectively cross-check the PMI for every possible $x \in X$ for each item in the stream. In practice, this is far too expensive and will lead to online algorithms that cannot keep up with the rate at which the input data is produced. To solve this problem, we propose a heuristic algorithm that sacrifices some accuracy for speed in computation.

Besides keeping processing times in check, we have to be careful about the memory requirements of any proposed algorithm. Recall that we are interested in retaining information for all pairs of $x$ and $y$, where each is drawn from a set of cardinality in the millions. Our algorithm uses approximate counting to retain the counts of all pairs of items $\langle x, y \rangle$ in a data structure $C_{xy}$. We keep exact counts of all $x$ and $y$ since this takes considerably less space. Given these values, we can (approximately) estimate $\mathrm{PMI}(x, y)$ for any $\langle x, y \rangle$ in the stream.

We assume $C_{xy}$ to be based on recent work in space efficient counting methods for streamed text data [Talbot, 2009; Van Durme and Lall, 2009]. For our implementation we used TOMB counters [Van Durme and Lall, 2009] which approximate counts by storing values in log-scale. These log-scale counts are maintained in unary within layers of Bloom filters [Bloom, 1970] (Figure 1) that can be probabilistically updated using a small base (Figure 2); each occurrence of an item in the stream prompts a probabilistic update to its value, dependent on the base. By tuning this base, one can trade off between the accuracy of the counts and the space savings of approximate counting.

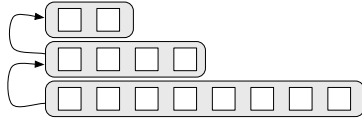

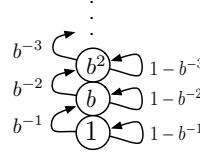

Figure 1: Unary counting with Bloom filters.     Figure 2: Transition by base $b$.

Now, to get around the problem of having stale PMI values because the count of $x$ changing (i.e., the issue in Example 2 in the previous section), we divide the stream up into fixed-size buffers $B$ and re-compute the PMIs for all pairs seen within each buffer (see Algorithm 1).

Updating counts for $x$, $y$ and $\langle x, y \rangle$ is constant time per element in the stream. Insertion into a $k$-best priority queue requires $O(\lg k)$ operations. Per interval, we perform in the worst case one insertion per new element observed, along with one insertion for each element stored in the previous rank lists. As long as $|B| \geq |X|k$, updating rank lists costs $O(|B|\lg k)$ per interval.[5] The algorithm therefore requires $O(n + n\lg k) = O(n\lg k)$ time, where $n$ is the length of the stream. Note that when $|B| = n$ we have the standard offline method for computing PMI across $X$ and $Y$ (not withstanding approximate counters). When $|B| < |X|k$, we run afoul of the lower bound given by Theorem 2. Regarding space, $|I| \leq |B|$. A benefit of our algorithm is that this can be kept significantly smaller than $|X| \times |Y|$,[6] since in practice, $|Y| \gg \lg k$.

**Algorithm 1** FIND-ONLINE-PMI

---

 1: initialize hashtable counters $H_x$ and $H_y$ for exact counts
 2: initialize an approximate counter $C_{xy}$
 3: initialize rank lists, $L$, mapping $x$ to $k$-best priority queue storing $\langle y, \mathrm{PMI}(x, y) \rangle$
 4: **for** each buffer $B$ in the stream **do**
 5:     initialize $I$, mapping $\langle x, y \rangle$ to $\{0, 1\}$, denoting whether $\langle x, y \rangle$ was observed in $B$
 6:     **for** $\langle x, y \rangle$ in $B$ **do**
 7:         set $I(\langle x, y \rangle) = 1$
 8:         increment $H_x(x)$ ▷ *initial value of 0*
 9:         increment $H_y(y)$ ▷ *initial value of 0*
10:         insert $\langle x, y \rangle$ into $C_{xy}$
11:     **end for**
12:     **for** each $x \in X$ **do**
13:         re-compute $L(x)$ using current $y \in L(x)$ and $\{y | I(\langle x, y \rangle) = 1\}$
14:     **end for**
15: **end for**

---

## 5.1 Misclassification Probability Bound

Our algorithm removes problems due to the count of $x$ changing, but does not solve the problem that the probability of $y$ changes (i.e., the issue in Example 1 in the previous section). The PMI of a pair $\langle x, y \rangle$ may decrease considerably if there are many occurrences of $y$ (and relatively few occurrences of $\langle x, y \rangle$) in the stream, leading to the removal of $y$ from the true top-$k$ list for $x$. We show in the following that this is not likely to happen very often for the text data that our algorithm is designed to work on.

In giving a bound on this error, we will make two assumptions: (i) the PMI for a given $x$ follows a Zipfian distribution (something that we observed in our data), and (ii) the items in the stream are drawn independently from some underlying distribution (i.e., they are i.i.d.). Both these assumptions together help us to sidestep the lower bound proved earlier and demonstrate that our single-pass algorithm will perform well on real language data sets.

We first make the observation that, for any $y$ in the set of top-$k$ PMIs for $x$, if $\langle x, y \rangle$ appears in the final buffer then we are guaranteed that $y$ is correctly placed in the top-$k$ at the end. This is because we recompute PMIs for all the pairs in the last buffer at the end of the algorithm (line 13 of Algorithm 1). The probability that $\langle x, y \rangle$ does not appear in the last buffer can be bounded using the i.i.d. assumption to be at most

$$\left( 1 - \frac{c(x, y)}{n} \right)^{|B|} \approx e^{-\frac{|B| c(x, y)}{n}}$$

$$\leq e^{-k |X| c(x, y) / n},$$

where for the last inequality we use the bound $|B| \geq |X| k$ that we assumed in the previous section. Hence, in those cases that $c(x, y) = \Omega(n / (|X| k))$, our algorithm correctly identifies $y$ as being in the top-$k$ PMI for $x$ with high probability. The proof for general $c(x, y)$ is given next.

We study the probability with which some $y'$ which is not in the top-$k$ PMI for a fixed $x$ can displace some $y$ in the top-$k$ PMI for $x$. We do so by studying the last buffer in which $\langle x, y \rangle$ appears. The only way that $y'$ can displace $y$ in the top-$k$ for $x$ in our algorithm is if at the end of this buffer the following holds true:

$$\frac{c_t(x, y')}{c_t(y')} > \frac{c_t(x, y)}{c_t(y)},$$

where the $t$ subscripts denotes the respective counts at the end of the buffer. We will show that this event occurs with very small probability. We do so by bounding the probability of the following three unlikely events.

If we assume all $c(x, y)$ are above some threshold $m$, then with only small probability (i.e., $1/2^m$) will the last buffer containing $\langle x, y \rangle$ appear before the midpoint of the stream. So, let us assume that the buffer appears after the midpoint of the stream. Then, the probability that $\langle x, y' \rangle$ appears more than $(1 + \delta) c(x, y') / 2$ times by this point can be bounded by the Chernoff bound to be at most

$\exp(-c(x, y')\delta^2/8)$. Similarly, the probability that $y'$ appears less than $(1-\delta)c(y')/2$ times by this point can be bounded by $\exp(-c(y')\delta^2/4)$. Putting all these together, we get that

$$\Pr\left(\frac{c_t(x, y')}{c_t(y')} > \frac{(1+\delta)c(x, y')}{(1-\delta)c(y')}\right) < 1/2^m + \exp(-c(x, y')\delta^2/8) + \exp(-c(y')\delta^2/4).$$

We now make use of the assumption that the PMIs are distributed in a Zipfian manner. Let us take the rank of the PMI of $y'$ to be $i$ (and recall that the rank of the PMI of $y$ is at most $k$). Then, by the Zipfian assumption, we have that $\mathrm{PMI}(x, y) \geq (i/k)^s \mathrm{PMI}(x, y')$, where $s$ is the Zipfian parameter. This can be re-written as $\frac{c(x,y)}{c(y)} \geq \frac{c(x,y')}{c(y')} 2^{((i/k)^s - 1)\mathrm{PMI}(x,y')}$. We can now put all these results together to bound the probability of the event

$$\Pr\left(\frac{c_t(x, y')}{c_t(y')} > \frac{c_t(x, y)}{c_t(y)}\right) \leq 1/2^m + \exp(-c(x, y')\delta^2/8) + \exp(-c(y')\delta^2/4),$$

where we take $\delta = (((i/k)^s - 1)2^{\mathrm{PMI}(x,y')} - 1)/(((i/k)^s - 1)2^{\mathrm{PMI}(x,y')} + 1)$.

Hence, the probability that some low-ranked $y'$ will displace a $y$ in the top-$k$ PMI of $x$ is low. Taking a union bound across all possible $y' \in Y$ gives a bound of $1/2^m + |Y|(\exp(-c(x, y')\delta^2/8) + \exp(-c(y')\delta^2/4))$.[7]

## 6 Experiments

We evaluated our algorithm for online, $k$-best PMI with a set of experiments on collecting *verbal triggers* in a document collection. For each document, we considered all verb::verb pairs, non-stemmed; e.g., *wrote::ruled*, *fighting::endure*, *argued::bore*. For each unique verb $x$ observed in the stream, our goal was to recover the top-$k$ verbs $y$ with the highest PMI given $x$.[8] Readers may peek ahead to Table 2 for example results.

Experiments were based on 100,000 NYTimes articles taken from the Gigaword Corpus [Graff, 2003]. Tokens were tagged for part of speech (POS) using SVMTool [Giménez and Màrquez, 2004], a POS tagger based on SVM$^{light}$ [Joachims, 1999].

Our stream was constructed by considering all pairwise combinations of the roughly 82 (on average) verb tokens occurring in each document. Where $D \in \mathcal{D}$ is a document in the collection, let $D_v$ refer to the list of verbal tokens, not necessarily unique. The length of our stream, $n$, is therefore: $\sum_{D \in \mathcal{D}} \binom{|D_v|}{2}$.[9]

While research into methods for space efficient, approximate counting has been motivated by a desire to handle exceptionally large datasets (using limited resources), we restricted ourselves here to a dataset that would allow for comparison to explicit, non-approximate counting (implemented through use of standard hashtables).[10] We will refer to such non-approximate counting as *perfect counting*. Finally, to guard against spurious results arising from rare terms, we employed the same $c(xy) > 5$ threshold as used by Church and Hanks [1990].

We did not heavily tune our counting mechanism to this task, other than to experiment with a few different bases (settling on a base of 1.25). As such, empirical results for approximate counting

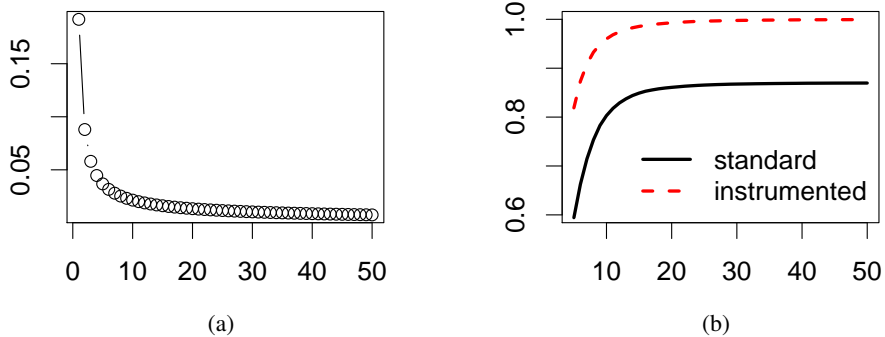

Figure 3: 3(a) : Normalized, mean PMI for top-50 $y$ for each $x$. 3(b) : Accuracy of top-5 ranklist using the *standard* measurement, and when using an *instrumented* counter that had oracle access to which $\langle x, y \rangle$ were above threshold.

Table 1: When using a perfect counter and a buffer of 50, 500 and 5,000 documents, for $k = 1, 5, 10$: the accuracy of the resultant $k$-best lists when compared to the first $k$, $k + 1$ and $k + 2$ true values.

| Buffer | 1 | 2 | 3 | 5 | 6 | 7 | 10 | 11 | 12 |
|---|---|---|---|---|---|---|---|---|---|
| 50 | 94.10 | 98.75 | 99.45 | 97.25 | 99.13 | 99.60 | 98.05 | 99.26 | 99.63 |
| 500 | 94.14 | 98.81 | 99.53 | 97.31 | 99.16 | 99.62 | 98.12 | 99.29 | 99.65 |
| 5000 | 94.69 | 98.93 | 99.60 | 97.76 | 99.30 | 99.71 | 98.55 | 99.46 | 99.74 |
| | $k = 1$ | | | $k = 5$ | | | $k = 10$ | | |

should be taken as a lower bound, while the perfect counting results are the upper bound on what an approximate counter might achieve.

We measured the accuracy of resultant $k$-best lists by first collecting the true top-50 elements for each $x$, offline, to be used as a key. Then, for a proposed $k$-best list, accuracy was calculated at different ranks of the gold standard. For example, the elements of a proposed 10-best list will optimally fully intersect with the first 10 elements of the gold standard. In the case the list is not perfect, we would hope that an element incorrectly positioned at, e.g., rank 9, should really be of rank 12, rather than rank 50.

Using this gold standard, Figure 3(a) shows the normalized, mean PMI scores as according to rank. This curve supports our earlier theoretical assumption that PMI over $Y$ is a Zipfian distribution for a given $x$.

## 6.1 Results

In Table 1 we see that when using a perfect counter, our algorithm succeeds in recovering almost all top-$k$ elements. For example, when $k = 5$, reading 500 documents at a time, our rank lists are $97.31\%$ accurate. Further, of those collected triggers that are not truly in the top-5, most were either in the top 6 or 7. As there appears to be minimal impact based on buffer size, we fixed $|B| = 500$ documents for the remainder of our experiments.[11] This result supports the intuition behind our misclassification probability bound: while it is possible for an adversary to construct a stream that would mislead our online algorithm, this seems to rarely occur in practice.

Shown in Figure 3(b) are the accuracy results when using an approximate counter and a buffer size of 500 documents, to collect top-5 rank lists. Two results are presented. The *standard* result is based on comparing the rank lists to the key just as with the results when using a perfect counter. A problem with this evaluation is that the hard threshold used for both generating the key, and the results for perfect counting, cannot be guaranteed to hold when using approximate counts. It is possible that

Table 2: Top 5 verbs, $y$, for $x = $ *bomb, laughed* and *vetoed*. Left columns are based on using a perfect counter, while right columns are based on an approximate counter. Numeral prefixes denote rank of element in true top-$k$ lists. All results are with respect to a buffer of 500 documents.

| $x = $ bomb | | $x = $ laughed | | $x = $ vetoed | |
|---|---|---|---|---|---|
| 1:detonate | 1:detonate | 1:tickle | -:panang | 1:vetoing | 1:vetoing |
| 2:assassinate | 7:bombed | 2:tickling | 1:tickle | 2:overridden | 2:overridden |
| 3:bomb | 2:assassinate | 3:tickled | 3:tickled | 3:overrode | 4:override |
| 4:plotting | 4:plotting | 4:snickered | 2:tickling | 4:override | 5:latches |
| 5:plotted | 8:expel | 5:captivating | 4:snickered | 5:latches | 7:vetoed |

some $\langle x, y \rangle$ pair that occurs perhaps 4 or 5 times may be misreported as occurring 6 times or more. In this case, the $\langle x, y \rangle$ pair will not appear in the key in any position, thus creating an artificial upper bound on the possible accuracy as according to this metric. For purposes of comparison, we *instrumented* the approximate solution to use a perfect counter in parallel. All PMI values were computed as before, using approximate counts, but the perfect counter was used just in verifying whether a given pair exceeded the threshold. In this way the approximate counting solution saw just those elements of the stream as observed in the perfect counting case, allowing us to evaluate the ranking error introduced by the counter, irrespective of issues in "dipping below" the threshold. As seen in the *instrumented* curve, top-5 rank lists generated when using the approximate counter are composed primarily of elements truly ranked 10 or below.

## 6.2 Examples

Figure 2 contains the top-5 most associated verbs as according to our algorithm, both when using a perfect and an approximate counter. As can be seen for the perfect counter, and as suggested by Table 1, in practice it is possible to track PMI scores over buffered intervals with a very high degree of accuracy. For the examples shown (and more generally throughout the results), the resultant $k$-best lists are near perfect matches to those computed offline.

When using an approximate counter we continue to see reasonable results, with some error introduced due to the use of probabilistic counting. The rank 1 entry reported for $x = $ *laughed* exemplifies the earlier referenced issue of the approximate counter being able to incorrectly dip below the threshold for terms that the gold standard would never see.[12]

## 7 Conclusions

In this paper we provided the first study of estimating top-$k$ PMI online. We showed that while a precise solution comes at a high cost in the streaming model, there exists a simple algorithm that performs well on real data. An avenue of future work is to drop the assumption that each of the top-$k$ PMI values is maintained explicitly and see whether there is an algorithm that is feasible for the streaming version of the problem or if a similar lower bound still applies. Another promising approach would be to apply the tools of two-way associations to this problem [Li and Church, 2007].

An experiment of Schooler and Anderson [1997] assumed words in NYTimes headlines operated as cues for the retrieval of memory structures associated with co-occurring terms. Missing from that report was how such cues might be accumulated over time. The work presented here can be taken as a step towards modeling resource constrained, online cue learning, where an appealing description of our model involves agents tracking co-occurring events over a local temporal window (such as a day), and regularly consolidating this information into long term memory (when they "sleep"). Future work may continue this direction by considering data from human trials.

**Acknowledgements**    Special thanks to Dan Gildea, as well as Rochester HLP/Jaeger-lab members for ideas and feedback. The first author was funded by a 2008 Provost's Multidisciplinary Award from the University of Rochester, and NSF grant IIS-0328849. The second author was supported in part by the NSF grants CNS-0905169 and CNS-0910592, funded under the American Recovery and Reinvestment Act of 2009 (Public Law 111-5), and by NSF grant CNS-0716423.

## Footnotes

[1] This situation holds in language modeling, such as in the context of machine translation.

[2] As is standard, lg refers to $\log_2$.

[3]Rosenfeld: ... *unlike in a bigram model, where the number of different consecutive word pairs is much less than [the vocabulary] $V^2$, the number of word pairs where both words occurred in the same document is a significant fraction of $V^2$.*

[4]Note that Frank *et al.* [2007] gave evidence suggesting PMI may be suboptimal for cue modeling, but to our understanding this result is limited to the case of novel language acquisition.

[5]I.e., the extra cost for reinserting elements from the previous rank lists is amortized over the buffer length.

[6]E.g., the $V^2$ of Rosenfeld.

[7]For streams composed such as described in our experiments, this bound becomes powerful as $m$ approaches 100 or beyond (recalling that both $c(x, y')$, $c(y') > m$). Experimentally we observed this to be conservative in that such errors appear unlikely even when using a smaller threshold (e.g., $m = 5$).

[8]Unlike in the case of Rosenfeld [1994], we allowed for triggers to occur anywhere in a document, rather than exclusively in the preceding context. This can be viewed as a restricted version of the experiments of Chambers and Jurafsky [2008], where we consider all verb pairs, regardless of whether they are assumed to possess a co-referent argument.

[9]For the experiments here, $n = 869,641,588$, or roughly 900 million, $\langle x, y \rangle$ pairs. If fully enumerated as text, this stream would have required 12GB of uncompressed storage. Vocabulary size, $|X| = |Y|$, was roughly 30 thousand (28,972) unique tokens.

[10]That is, since our algorithm is susceptible to adversarial manipulation of the stream, it is important to establish the experimental upper bound that is possible assuming zero error due to the use of probabilistic counts.

[11]Strictly speaking, $|B|$ is no larger than the maximum length interval in the stream resulting from enumerating the contents of, e.g., 500 consecutive documents.

[12]I.e., the token *panang*, incorrectly tagged as a verb, is sparsely occurring.

# References

[Bloom, 1970] Burton H. Bloom. Space/time trade-offs in hash coding with allowable errors. *Communications of the ACM*, 13:422–426, 1970.

[Chambers and Jurafsky, 2008] Nathanael Chambers and Dan Jurafsky. Unsupervised Learning of Narrative Event Chains. In *Proceedings of ACL*, 2008.

[Chklovski and Pantel, 2004] Timothy Chklovski and Patrick Pantel. VerbOcean: Mining the Web for Fine-Grained Semantic Verb Relations. In *Proceedings of Conference on Empirical Methods in Natural Language Processing (EMNLP-04)*, pages 33–40, Barcelona, Spain, 2004.

[Church and Hanks, 1990] Kenneth Church and Patrick Hanks. Word Association Norms, Mutual Information and Lexicography. *Computational Linguistics*, 16(1):22–29, March 1990.

[Frank *et al.*, 2007] Michael C. Frank, Noah D. Goodman, and Joshua B. Tenenbaum. A Bayesian framework for cross-situational word learning. In *Advances in Neural Information Processing Systems, 20*, 2007.

[Giménez and Màrquez, 2004] Jesús Giménez and Lluís Màrquez. SVMTool: A general POS tagger generator based on Support Vector Machines. In *Proceedings of LREC*, 2004.

[Graff, 2003] David Graff. English Gigaword. Linguistic Data Consortium, Philadelphia, 2003.

[Joachims, 1999] Thorsten Joachims. Making large-scale SVM learning practical. In B. Schölkopf, C. Burges, and A. Smola, editors, *Advances in Kernel Methods - Support Vector Learning*, chapter 11, pages 169–184. MIT Press, Cambridge, MA, 1999.

[Li and Church, 2007] Ping Li and Kenneth W. Church. A sketch algorithm for estimating two-way and multi-way associations. *Computational Linguistics*, 33(3):305–354, 2007.

[Lin, 1998] Dekang Lin. Automatic Retrieval and Clustering of Similar Words. In *Proceedings of COLING-ACL*, 1998.

[Rosenfeld, 1994] Ronald Rosenfeld. *Adaptive Statistical Language Modeling: A Maximum Entropy Approach*. PhD thesis, Computer Science Department, Carnegie Mellon University, April 1994.

[Schooler and Anderson, 1997] Lael J. Schooler and John R. Anderson. The role of process in the rational analysis of memory. *Cognitive Psychology*, 32(3):219–250, 1997.

[Talbot and Osborne, 2007] David Talbot and Miles Osborne. Randomised Language Modelling for Statistical Machine Translation. In *Proceedings of ACL*, 2007.

[Talbot, 2009] David Talbot. Succinct approximate counting of skewed data. In *Proceedings of IJCAI*, 2009.

[Van Durme and Lall, 2009] Benjamin Van Durme and Ashwin Lall. Probabilistic Counting with Randomized Storage. In *Proceedings of IJCAI*, 2009.

